# Classifying with Gaussian Mixtures and Clusters

**Nanda Kambhatla and Todd K. Leen**
Department of Computer Science and Engineering
Oregon Graduate Institute of Science & Technology
P.O. Box 91000 Portland, OR 97291-1000
*nanda@cse.ogi.edu, tleen@cse.ogi.edu*

## Abstract

In this paper, we derive classifiers which are *winner-take-all (WTA)* approximations to a Bayes classifier with Gaussian mixtures for class conditional densities. The derived classifiers include clustering based algorithms like LVQ and k-Means. We propose a *constrained rank Gaussian mixtures* model and derive a WTA algorithm for it. Our experiments with two speech classification tasks indicate that the constrained rank model and the WTA approximations improve the performance over the unconstrained models.

## 1 Introduction

A classifier assigns vectors from $\mathcal{R}^n$ ($n$ dimensional feature space) to one of $K$ classes, partitioning the feature space into a set of $K$ disjoint regions. A *Bayesian* classifier builds the partition based on a model of the class conditional probability densities of the inputs (the partition is optimal for the given model).

In this paper, we assume that the class conditional densities are modeled by mixtures of Gaussians. Based on Nowlan's work relating Gaussian mixtures and clustering (Nowlan 1991), we derive *winner-take-all (WTA)* algorithms which approximate a Gaussian mixtures Bayes classifier. We also show the relationship of these algorithms to non-Bayesian cluster-based techniques like LVQ and k-Means.

The main problem with using Gaussian mixtures (or WTA algorithms thereof) is the explosion in the number of parameters with the input dimensionality. We propose

a constrained rank Gaussian mixtures model for classification. Constraining the rank of the Gaussians reduces the effective number of model parameters thereby regularizing the model. We present the model and derive a WTA algorithm for it. Finally, we compare the performance of the different mixture models discussed in this paper for two speech classification tasks.

## 2  Gaussian Mixture Bayes (GMB) classifiers

Let $x$ denote the feature vector ($x \in \mathcal{R}^n$), and $\{\Omega^I, I = 1, \ldots, K\}$ denote the classes. Class priors are denoted $p(\Omega^I)$ and the class-conditional densities are denoted $p(x \mid \Omega^I)$. The discriminant function for the Bayes classifier is

$$\delta^I(x) = p(\Omega^I) \; p(x \mid \Omega^I) \,. \tag{1}$$

An input feature vector $x$ is assigned to class $I$ if $\delta^I(x) > \delta^J(x) \;\; \forall J \neq I$. Given the class conditional densities, this choice minimizes the classification error rate (Duda and Hart 1973).

We model each class conditional density by a mixture composed of $Q^I$ component Gaussians. The Bayes discriminant function (see Figure 1) becomes

$$\hat{\delta}^I(x) = p(\Omega^I) \sum_{j=1}^{Q^I} \frac{\alpha_j^I}{(2\pi)^{n/2} \sqrt{|\Sigma_j^I|}} \exp\left[ -\frac{1}{2}(x - \mu_j^I)^T {\Sigma_j^I}^{-1} (x - \mu_j^I) \right] \,, \tag{2}$$

where $\mu_j^I$ and $\Sigma_j^I$ are the mean and the covariance matrix of the $j^{th}$ mixture component for $\Omega^I$.

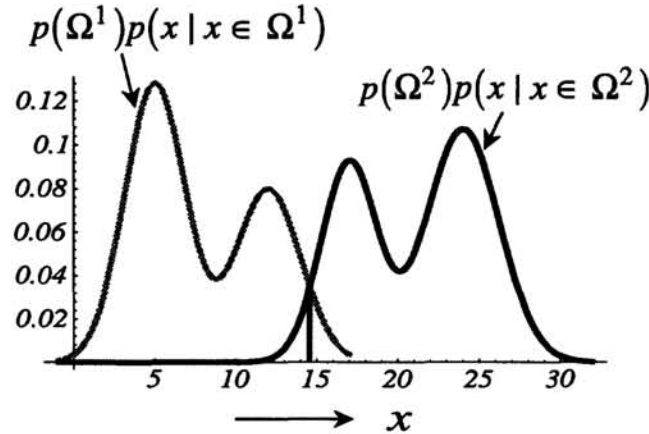

Fig. 1: Figure showing the decision rule of a GMB classifier for a two class problem with one input feature. The horizontal axis represents the feature and the vertical axis represents the Bayes discriminant functions. In this example, the class conditional densities are modelled as a mixture of two Gaussians and equal priors are assumed.

To implement the Gaussian mixture Bayes classifier (GMB) we first separate the training data into the different classes. We then use the EM algorithm (Dempster

*et al* 1977, Nowlan 1991) to determine the parameters for the Gaussian mixture density for each class.

# 3   Winner-take-all approximations to GMB classifiers

In this section, we derive winner-take-all (WTA) approximations to GMB classifiers. We also show the relationship of these algorithms to non-Bayesian cluster-based techniques like LVQ and k-Means.

## 3.1   The WTA model for GMB

The WTA assumptions (relating hard clustering to Gaussian mixtures; see (Nowlan 1991)) are:

- $p(x \mid \Omega^I)$ are mixtures of Gaussians as in (2).
- The summation in (2) is dominated by the largest term. This is "equivalent to assigning all of the responsibility for an observation to the Gaussian with the highest probability of generating that observation" (Nowlan 1991).

To draw the relation between GMB and cluster-based classifiers, we further assume that:

- The mixing proportions $(\alpha_j^I)$ are equal for a given class.
- The number of mixture components $Q^I$ is proportional to $p(\Omega^I)$.

Applying all the above assumptions to (2), taking logs and discarding the terms that are identical for each class, we get the discriminant function

$$\hat{\gamma}^I(x) = - \min_{j=1}^{Q^I} \left[ \frac{1}{2}\log(|\Sigma_j^I|) + \frac{1}{2}(x - \mu_j^I)^T \Sigma_j^{I^{-1}} (x - \mu_j^I) \right] . \qquad (3)$$

The discriminant function (3) suggests an algorithm that approximates the Bayes classifier. We segregate the feature vectors by class and then train a separate vector quantizer (VQ) for each class. We then compute the means $\mu_j^I$ and the covariance matrices $\Sigma_j^I$ for each Voronoi cell of each quantizer, and use (3) for classifying new patterns. We call this algorithm **VQ-Covariance**. Note that this algorithm *does not* do a maximum likelihood estimation of its parameters based on the probability model used to derive (3). The probability model is only used to classify patterns.

## 3.2   The relation to LVQ and k-Means

Further assume that for each class, the mixture components are spherically symmetric with covariance matrix $\Sigma_j^I = \sigma^2 I$, with $\sigma^2$ identical for all classes. We obtain the discriminant function,

$$\hat{\gamma}^I(x) = - \min_{j=1}^{Q^I} \| x - \mu_j^I \|^2 . \qquad (4)$$

This is exactly the discriminant function used by the learning vector quantizer (LVQ; Kohonen 1989) algorithm. Though LVQ employs a discriminatory training procedure (i.e it directly learns the class boundaries and does not *explicitly* build a separate model for each class), the implicit model of the class conditional densities used by LVQ corresponds to a GMB model under all the assumptions listed above. This is also the implicit model underlying any classifier which makes its classification decision based on the Euclidean distance measure between a feature vector and a set of prototype vectors (e.g. a k-Means clustering followed by classification based on (4)).

# 4   Constrained rank GMB classifiers

In the preceding sections, we have presented a GMB classifier and some WTA approximations to GMB. Mixture models such as GMB generally have too many parameters for small data sets. In this section, we propose a way of regularizing the mixture densities and derive a WTA classifier for the regularized model.

## 4.1   The constrained rank model

In section 2, we assumed that the class conditional densities of the feature vectors $x$ are mixtures of Gaussians

$$
\begin{aligned}
p(x \mid \Omega^I) &= \sum_{j=1}^{Q^I} \frac{\alpha_j^I}{(2\pi)^{n/2}\sqrt{|\Sigma_j^I|}} \exp\left[-\frac{1}{2}(x-\mu_j^I)^T \Sigma_j^{I^{-1}}(x-\mu_j^I)\right] \\
&= \sum_{j=1}^{Q^I} \frac{\alpha_j^I}{(2\pi)^{n/2}\sqrt{\prod_{i=1}^n \lambda_{ji}^I}} \exp\left[-\frac{1}{2}(x-\mu_j^I)^T \left(\sum_{i=1}^n \frac{e_{ji}^I e_{ji}^{I\,T}}{\lambda_{ji}^I}\right)(x-\mu_j^I)\right]
\end{aligned}
$$

$$(5)$$

where $\mu_j^I$ and $\Sigma_j^I$ are the means and covariance matrices for the $j^{th}$ component Gaussian. $e_{ji}^I$ and $\lambda_{ji}^I$ are the orthonormal eigenvectors and eigenvalues of $\Sigma_{ji}^I$ (ordered such that $\lambda_{j1}^I \geq \ldots \geq \lambda_{jn}^I$). In (5), we have written the Mahalanobis distance in terms of the eigenvectors.

For a particular data point $x$, the Mahalanobis distance is very sensitive to changes in the squared projections onto the *trailing* eigen-directions, since the variances are very small in these directions. This is a potential problem with small data sets. When there are insufficient data points to estimate all the parameters of the mixture density accurately, the trailing eigen-directions and their associated eigenvalues are likely to be poorly estimated. Using the Mahalanobis distance in (5) can lead to erroneous results in such cases.

We propose a method for regularizing Gaussian mixture classifiers based on the above ideas. We assume that the trailing $n - m$ eigen-directions of each Gaussian component are inaccurate due to overfitting to the training set. We rewrite the class conditional densities (5) retaining only the leading $m$ ($0 < m \leq n$) eigen-directions

in the determinants and the Mahalanobis distances

$$p(x \mid \Omega^I) = \sum_{j=1}^{Q^I} \frac{\alpha_j^I}{(2\pi)^{m/2} \sqrt{\prod_{i=1}^m \lambda_{ji}^I}} \exp\left[ -\frac{1}{2}(x - \mu_j^I)^T \left( \sum_{i=1}^m \frac{e_{ji}^I e_{ji}^I{}^T}{\lambda_{ji}^I} \right) (x - \mu_j^I) \right] .$$

(6)

We choose the value of $m$ (the reduced rank) by cross-validation over a separate validation set. Thus, our model can be considered to be regularizing or constraining the class conditional mixture densities.

If we apply the above model and derive the Bayes discriminant functions (1), we get,

$$\hat{\delta}^I(x) = p(\Omega^I) \sum_{j=1}^{Q^I} \frac{\alpha_j^I}{(2\pi)^{m/2} \sqrt{\prod_{i=1}^m \lambda_{ji}^I}} \exp\left[ -\frac{1}{2}(x - \mu_j^I)^T \left( \sum_{i=1}^m \frac{e_{ji}^I e_{ji}^I{}^T}{\lambda_{ji}^I} \right) (x - \mu_j^I) \right] .$$

(7)

We can implement a constrained rank Gaussian mixture Bayes (GMB-Reduced) classifier based on (7) using the EM algorithm to determine the parameters of the mixture density for each class. We segregate the data into different classes and use the EM algorithm to determine the parameters of the full mixture density (5). We then use (7) to classify patterns.

## 4.2 A constrained rank WTA algorithm

We now derive a winner-take-all (WTA) approximation for the constrained rank mixture model described above. We assume (similar to section 3.1) that

- $p(x \mid \Omega^I)$ are constrained mixtures of Gaussians as in (6).
- The summation in (6) is dominated by the largest term (the *WTA* assumption).
- The mixing proportions ($\alpha_j^I$) are equal for a given class and the number of components $Q^I$ is proportional to $p(\Omega^I)$.

Applying these assumptions to (7), taking logs and discarding the terms that are identical for each class, we get the discriminant function

$$\hat{\gamma}^I(x) = -\min_{j=1}^{Q^I} \left[ \frac{1}{2} \sum_{i=1}^m \log(\lambda_{ji}^I) + \frac{1}{2}(x - \mu_j^I)^T \left( \sum_{i=1}^m \frac{e_{ji}^I e_{ji}^I{}^T}{\lambda_{ji}^I} \right) (x - \mu_j^I) \right] .$$

(8)

It is interesting to compare (8) with (3). Our model postulates that the trailing $n - m$ eigen-directions of each Gaussian represent overfitting to noise in the training set. The discriminant functions reflect this; (8) retains only those terms of (3) which are in the leading $m$ eigen-directions of each Gaussian.

We can generate an algorithm based on (8) that approximates the reduced rank Bayes classifier. We separate the data based on classes and train a separate vector quantizer (VQ) for each class. We then compute the means $\mu_j^I$, the covariance matrices $\Sigma_j^I$ for each Voronoi cell of each quantizer and the orthonormal eigenvectors

Table 1: The **test set** classification accuracies for the TIMIT vowels data for different algorithms.

| ALGORITHM | ACCURACY |
|---|---|
| MLP (40 nodes in hidden layer) | 46.8% |
| GMB (1 component; full) | 41.4% |
| GMB (1 component; diagonal) | 46.3% |
| GMB-Reduced (1 component; 13-D) | 51.2% |
| VQ-Covariance (1 component) | 41.4% |
| VQ-Covariance-Reduced (1 component; 13-D) | 51.2% |
| LVQ (48 cells) | 41.4% |

$e^I_j i$ and eigenvalues $\lambda^I_j$ for each covariance matrix $\Sigma^I_j$. We use (8) for classifying new patterns. Notice that the algorithm described above is a reduced rank version of VQ-Covariance (described in section 3.1). We call this algorithm **VQ-Covariance-Reduced**.

## 5   Experimental Results

In this section we compare the different mixture models and a multi layer perceptron (MLP) for two speech phoneme classification tasks. The measure used is the classification accuracy.

### 5.1   TIMIT data

The first task is the classification of 12 monothongal vowels from the TIMIT database (Fisher and Doddington 1986). Each feature vector consists of the lowest 32 DFT coefficients, time-averaged over the central third of the vowel. We partitioned the data into a training set (1200 vectors), a validation set (408 vectors) for model selection, and a test set (408 vectors). The training set contained 100 examples of each class. The values of the free parameters for the algorithms (the number of component densities, number of hidden nodes for the MLP etc.) were selected by maximizing the performance on the validation set.

Table 1 shows the results obtained with different algorithms. The constrained rank models (GMB-Reduced and VQ-Covariance-Reduced[1]) perform much better than all the unconstrained ones and even beat a MLP for this task. This data set consists of very few data points per class, and hence is particularly susceptible to overfitting by algorithms with a large number of parameters (like GMB). It is not surprising that constraining the number of model parameters is a big win for this task.

Table 2: The **test set** classification accuracies for the CENSUS data for different algorithms.

| ALGORITHM | ACCURACY |
|---|---|
| MLP (80 nodes in hidden layer) | 88.2% |
| GMB (1 component; full) | 77.2% |
| GMB (8 components; diagonal) | 70.9% |
| GMB-Reduced (2 components; 35-D) | 82.5% |
| VQ-Covariance (3 components) | 77.5% |
| VQ-Covariance-Reduced (4 components; 38-D) | 84.2% |
| LVQ (55 cells) | 67.3% |

## 5.2 CENSUS data

The next task we experimented with was the classification of 9 vowels (found in the utterances of the days of the week). The data was drawn from the CENSUS speech corpus (Cole *et al* 1994). Each feature vector was 70 dimensional (perceptual linear prediction (PLP) coefficients (Hermansky 1990) over the vowel and surrounding context). We partitioned the data into a training set (8997 vectors), a validation set (1362 vectors) for model selection, and a test set (1638 vectors). The training set had close to a 1000 vectors per class. The values of the free parameters for the different algorithms were selected by maximizing the validation set performance.

Table 2 gives a summary of the classification accuracies obtained using the different algorithms. This data set has a lot more data points per class than the TIMIT data set. The best accuracy is obtained by a MLP, though the constrained rank mixture models still greatly outperform the unconstrained ones.

## 6 Discussion

We have derived WTA approximations to GMB classifiers and shown their relation to LVQ and k-Means algorithms. The main problem with Gaussian mixture models is the explosion in the number of model parameters with input dimensionality, resulting in poor generalization performance. We propose *constrained rank* Gaussian mixture models for classification. This approach ignores some directions (*"noise"*) locally in the input space, and thus reduces the effective number of model parameters. This can be considered as a way of regularizing the mixture models. Our results with speech vowel classification indicate that this approach works better than using full mixture models, especially when the data set size is small.

The WTA algorithms proposed in this paper do not perform a maximum likelihood estimation of their parameters. The probability model is only used to classify data. We can potentially improve the performance of these algorithms by doing maximum likelihood training with respect to the models presented here.

**Acknowledgments**

This work was supported by grants from the Air Force Office of Scientific Research (F49620-93-1-0253), Electric Power Research Institute (RP8015-2) and the Office of Naval Research (N00014-91-J-1482). We would like to thank Joachim Utans, OGI for several useful discussions and Zoubin Ghahramani, MIT for providing MATLAB code for the EM algorithm. We also thank our colleagues in the Center for Spoken Language Understanding at OGI for providing speech data.

**References**

R.A. Cole, D.G. Novick, D. Burnett, B. Hansen, S. Sutton, M. Fanty. (1994) Towards Automatic Collection of the U.S. Census. *Proceedings of the International Conference on Acoustics, Speech and Signal Processing 1994.*

A.P. Dempster, N.M. Laird, and D.B. Rubin. (1977) Maximum Likelihood from Incomplete Data via the EM Algorithm. *J. Royal Statistical Society Series B*, vol. 39, pp. 1-38.

R.O. Duda and P.E. Hart. (1973) Pattern Classification and Scene Analysis. John Wiley and Sons Inc.

W.M Fisher and G.R Doddington. (1986) The DARPA speech recognition database: specification and status. In *Proceedings of the DARPA Speech Recognition Workshop*, p93-99, Palo Alto CA.

H. Hermansky. (1990) Perceptual Linear Predictive (PLP) analysis of speech. *J. Acoust. Soc. Am.*, 87(4):1738-1752.

T. Kohonen. (1989) Self-Organization and Associative Memory (3rd edition). Berlin: Springer-Verlag.

S.J. Nowlan. (1991) Soft Competitive Adaptation: Neural Network Learning Algorithms based on Fitting Statistical Mixtures. CMU-CS-91-126 PhD thesis, School of Computer Science, Carnegie Mellon University.

## Footnotes

[1]Note that since the best validation set performance is obtained with only one component for each mixture density, the WTA algorithms are identical to the GMB algorithms (for these results).
